# The RA Scanner: Prediction of Rheumatoid Joint Inflammation Based on Laser Imaging

**Anton Schwaighofer**[1,2]
[1] TU Graz, Institute for Theoretical Computer Science
Inffeldgasse 16b, 8010 Graz, Austria
`http://www.igi.tugraz.at/aschwaig`

**Volker Tresp, Peter Mayer**
[2] Siemens Corporate Technology, Department of Neural Computation
Otto-Hahn-Ring 6, 81739 Munich, Germany
`http://www.tresp.org,peter.mayer@mchp.siemens.de`

**Alexander K. Scheel, Gerhard Müller**
University Göttingen, Department of Medicine, Nephrology and Rheumatology
Robert-Koch-Straße 40, 37075 Göttingen, Germany
`ascheel@gwdg.de,gmueller@med.uni-goettingen.de`

## Abstract

We describe the RA scanner, a novel system for the examination of patients suffering from rheumatoid arthritis. The RA scanner is based on a novel laser-based imaging technique which is sensitive to the optical characteristics of finger joint tissue. Based on the laser images, finger joints are classified according to whether the inflammatory status has improved or worsened. To perform the classification task, various linear and kernel-based systems were implemented and their performances were compared. Special emphasis was put on measures to reliably perform parameter tuning and evaluation, since only a very small data set was available. Based on the results presented in this paper, it was concluded that the RA scanner permits a reliable classification of pathological finger joints, thus paving the way for a further development from prototype to product stage.

## 1 Introduction

Rheumatoid arthritis (RA) is the most common inflammatory arthropathy with 1–2% of the population being affected. This chronic, mostly progressive disease often leads to early disability and joint deformities. Recent studies have convincingly shown that early treatment and therefore an early diagnosis is mandatory to prevent or at least delay joint destruction [2]. Unfortunately, long-term medication with disease modifying anti-rheumatic drugs (DMARDs) often acts very slowly on clinical parameters of inflammation, making it difficult to find the right drug for a patient within adequate time. Conventional radiology,

such as magnetic resonance imaging (MRI) and ultrasound, may provide information on soft tissue changes, yet these techniques are time-consuming and—in the case of MRI—costly. New imaging techniques for RA diagnosis should thus be non-invasive, of low cost, examiner independent and easy to use.

Following recent experiments on absorption and scattering coefficients of laser light in joint tissue [6], a prototype laser imaging technique was developed [7]. As part of the prototype development, it became necessary to analyze if the rheumatic status of a finger joint can be reliably classified on the basis of the laser images. Aim of this article is to provide an overview of this analysis. Employing different linear and kernel-based classifiers, we will investigate the performance of the laser imaging technique to predict the status of the rheumatic joint inflammation. Provided that the accuracy of the overall system is sufficiently high, the imaging technique and the automatic inflammation classification can be combined into a novel device that allows an inexpensive and objective assessment of inflammatory joint changes.

The paper is organized as follows. In Sec. 2 we describe the RA scanner in more detail, as well as the process of data acquisition. In Sec. 3 we describe the linear and kernel-based classifiers used in the experiments. In Sec. 4 we describe how the methods were evaluated and compared. We present experimental results in Sec. 5. Conclusions and an outlook are given in Sec. 6.

## 2 The RA Scanner

The rheumatoid arthritis (RA) scanner provides a new medical imaging technique, developed specifically for the diagnosis of RA in finger joints. The RA scanner [7] allows the *in vivo* trans-illumination of finger joints with laser light in the near infrared wavelength range. The scattered light distribution is detected by a camera and is used to assess the inflammatory status of the finger joint. Example images, taken from an inflamed joint and from a healthy control, are shown in Fig. 1.

Starting out from the laser images, image pre-processing is used to obtain a description of each laser image by nine numerical features. A brief description of the features is given in Fig. 1. Furthermore for each finger joint examined, the circumference is measured using a conventional measuring tape. The nine image features plus the joint circumference make up the data that is used in the classification step of the RA scanner to predict the inflammatory status of the joint.

### 2.1 Data Acquisition

One of the clinically important questions is to know as early as possible if a prescribed medication improves the state of rheumatoid arthritis. Therefore the goal of the classification step in the RA scanner is to decide—based on features extracted from the laser images—if there was an improvement of arthritis activity or if the joint inflammation remained unchanged or worsened.

The data for the development of the RA scanner stems from a study on 22 patients with rheumatoid arthritis. Data from 72 finger joints were used for the study. All of these 72 finger joints were examined at baseline and during a follow-up visit after a mean duration of 42 days. Earlier data from an additional 20 patients had to be discarded since experimental conditions were not controlled properly.

Each joint was examined and the clinical arthritis activity was classified from 0 (inactive, not swollen, tender or warm) to 3 (very active) by a rheumatologist. The characteristics of joint tissue was recorded by the above described laser imaging technique. In a preprocess-

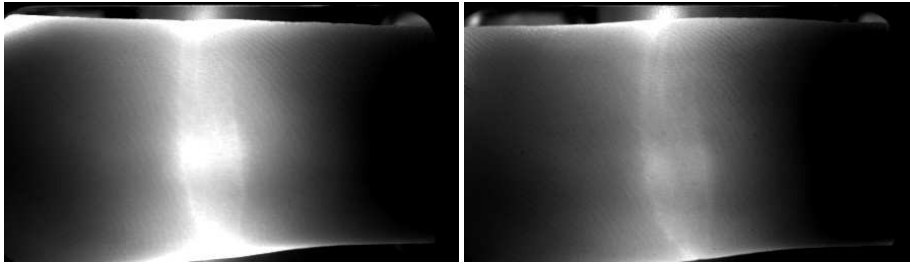

(a) Laser image of a healthy finger joint

(b) Laser image of an inflamed finger joint. The inflammation changes the joint tissue's absorption coefficient, giving a darker image.

Figure 1: Two examples of the light distribution captured by the RA scanner. A laser beam is sent through the finger joint (the finger tip is to the right, the palm is on the left), the light distribution below the joint is captured by a CCD element. To calculate the features, first a horizontal line near the vertical center of the finger joint is selected. The distribution of light intensity along that line is bell-shaped. The features used in the classification task are the maximum light intensity, the curvature of the light intensity at the maximum and seven additional features based on higher moments of the intensity curve.

ing step nine features were derived from the distribution of the scattered laser light (see Fig. 1). The tenth feature is the circumference of the finger joint.

Since there are high inter-individual variations in optical joint characteristics, it is not possible to tell the inflammatory status of a joint from one single image. Instead, special emphasis was put on the intra-individual comparison of baseline and follow-up data. For every joint examined, data from baseline and follow-up visit were compared and changes in arthritis activity were rated as improvement, unchanged or worsening.

This rating divided the data into two classes: Class $+1$ contains the joints where an improvement of arthritis activity was observed (a total of 46 joints), and class $-1$ are the joints that remained unchanged or worsened (a total of 26 joints). For all joints, the differences in feature values between baseline and follow-up visit were computed.

## 3 Classification Methods

In this section, we describe the employed linear and kernel-based classification methods, where we focus on design issues.

### 3.1 Gaussian Process Classification (GPC)

In Gaussian processes, a function

$$f(\mathbf{x}) = \sum_{j=1}^{M} w_j K(\mathbf{x}, \mathbf{x}_j, \Theta) \tag{1}$$

is described as a superposition of $M$ kernel functions $K(\mathbf{x}, \mathbf{x}_j, \Theta)$, defined for each of the $M$ training data points $\mathbf{x}_j$, with weight $w_j$. The kernel functions are parameterized by the vector $\Theta = (\theta_0 \dots \theta_d)$. In two-class Gaussian process classification, the logistic transfer function $\sigma(f(\mathbf{x})) = (1 + e^{-f(\mathbf{x})})^{-1}$ is applied to the prediction of a Gaussian process to produce an output which can be interpreted as $\pi(\mathbf{x})$, the probability of the input $\mathbf{x}$ belonging to class 1 [10].

In the experiment we chose the Gaussian kernel function

$$K(\mathbf{x}, \mathbf{x}_j, \Theta) = \theta_0 \exp\left[-\frac{1}{2}(\mathbf{x} - \mathbf{x}_j)^T \operatorname{diag}(\theta_1^2 \ldots \theta_d^2)^{-1}(\mathbf{x} - \mathbf{x}_j)\right] \qquad (2)$$

with input length scales $\theta_1 \ldots \theta_d$ where $d$ is the dimension of the input space. $\operatorname{diag}(\theta_1^2 \ldots \theta_d^2)$ denotes a diagonal matrix with entries $\theta_1^2 \ldots \theta_d^2$. For training the Gaussian process classifier (that is, determining the posterior probabilities of the parameters $w_1, \ldots w_M, \theta_0, \ldots \theta_d$) we used a full Bayesian approach, implemented with Readford Neal's freely available FBM software.[1]

## 3.2 Gaussian Process Regression (GPR)

In GPR we treat the classification problem as a regression problem with target values $\{-1, +1\}$, i.e. we do not apply the logistic transfer function as in the last subsection. Any GP output $< 0$ is treated as indicating an example from class 0, any output $>= 0$ as an indicator for class 1.The disadvantage is that the GPR prediction cannot be treated as a posterior class probability; the advantage is that the fast and non-iterative training algorithms for GPR can be applied. GPR for classification problems can be considered as special cases of Fisher discriminant analysis with kernels [4] and of least squares support vector machines [9].

The parameters $\Theta = \{\theta_0 \ldots \theta_d\}$ of the covariance function Eq. (2) were chosen by maximizing the posterior probability of $\Theta$, $P(\Theta|\mathbf{t}, X) \propto P(\mathbf{t}|X, \Theta)P(\Theta)$, via a scaled conjugate gradient method. Later on, this method will be referred to as "GPR Bayesian". Results are also given for a simplified covariance function with $\theta_0 = 1$, $\theta_1 = \theta_2 = \ldots = \theta_d = r$, where the common length scale $r$ was chosen by cross-validation (later on referred to as "GPR crossval").

## 3.3 Support Vector Machine (SVM)

The SVM is a maximum margin linear classifier. As in Sec. 3.2, the SVM classifies a pattern according to the sign of $f(x)$ in Eq. (1). The difference is that the weights $\mathbf{w} = (w_1, \ldots, w_M)^T$ in the SVM minimize the particular cost function [8]

$$\mathbf{w}^T \mathbf{K} \mathbf{w} + \sum_{i=1}^{M} C_i (1 - y_i(f(\mathbf{x}_i)))_+ \qquad (3)$$

where $()_+$ sets all negative arguments to zero. Here, $y_i \in \{+1, -1\}$ is the class label for training point $\mathbf{x}_i$. $C_i \geq 0$ is a constant that determines the weight of errors on the training data, and $\mathbf{K}$ is an $M \times M$ matrix containing the amplitudes of the kernel functions at the training data, i.e. $\mathbf{K}_{i,j} = K(\mathbf{x}_i, \mathbf{x}_j, \Theta)$. The motivation for this cost function stems from statistical learning theory [8]. Many authors have previously obtained excellent classification results by using the SVM. One particular feature of the SVM is the sparsity of the solution vector $\mathbf{w}$, that is, many elements $w_i$ are zero.

In the experiments, we used both an SVM with linear kernel ("SVM linear") and an SVM with a Gaussian kernel ("SVM Gaussian"), equivalent to the Gaussian process kernel Eq. (2), with $\theta_0 = 1$, $\theta_1 = \theta_2 = \ldots = \theta_d = r$. The kernel parameter $r$ was chosen by cross-validation.

To compensate for the unbalanced distribution of classes, the penalty term $C_i$ was chosen to be 0.8 for the examples from the larger class and 1 for the smaller class. This was found empirically to give the best balance of sensitivity and specificity (cf. Sec. 4). A formal treatment of this issue can be found in Lin et al. [3].

### 3.4 Generalized Linear Model (GLM)

A GLM for binary responses is built up from a linear model for the input data, and the model output $f(\mathbf{x}) = \mathbf{w}^T \mathbf{x}$ is in turn input to the link function. For Bernoulli distributions, the natural link function [1] is the logistic transfer function $\sigma(f(\mathbf{x})) = (1 + e^{-f(\mathbf{x})})^{-1}$. The overall output of the GLM $\sigma(f(\mathbf{x}))$ computes $\pi(\mathbf{x})$, the probability of the input $\mathbf{x}$ belonging to class 1. Training of the linear model was done by iteratively re-weighted least squares (IRLS).

## 4 Training and Evaluation

One of the challenges in developing the classification system for the RA scanner is the low number of training examples available. Data was collected through an extensive medical study, but only data from 72 fingers were found to be suitable for further use. Further data can only be acquired in carefully controlled future studies, once the initial prototype method has proven sufficiently successful.

**Training**    From the currently available 72 training examples, classifiers need to be trained and evaluated reliably. Part of the standard methodology for small data sets is N-fold cross-validation, where the data are partitioned into $N$ equally sized sets and the system is trained on $N-1$ of those sets and tested on the $N$th data set left out. Since we wish to make use of as much training data as possible, $N = 36$ seemed the appropriate choice [2], giving test sets with two examples in each iteration. For some of the methods model parameter needed to be tuned (for example, choosing SVM kernel width), where again cross-validation is employed. The nested cross-validation ensures that in no case any of the test examples is used for training or to tune parameters, leading to the following procedure:

```
Run 36 fold CV
  For Bayesian methods or methods without tunable parameters
      (SVM linear, GPC, GPR Bayesian, GLM):
      Use full training set to tune and train classifier
  For Non-Bayesian methods (SVM Gaussian, GPR crossval):
      Run 35 fold CV on the training set
          choose parameters to minimise CV error
      train classifier with chosen parameters
  evaluate the classifier on the 2 example test set
```

**Significance Tests**    In order to compare the performance of two given classification methods, one usually employs statistical hypothesis testing. We use here a test that is best suited for small test sets, since it takes into account the outcome on the test examples one by one, thus matching our above described 36-fold cross validation scheme perfectly. A similar test has been used by Yang and Liu [11] to compare text categorization methods.

Basis of the test are two counts $b$ (how many examples in the test set were correctly classified by method B, but misclassified by method A) and $c$ (number of examples misclassified by B, correctly classified by A). We assume that examples misclassified (resp. correctly classified) by both A and B do not contribute to the performance difference. We take the

| Method | Error rate |
|---|---|
| GLM | 20.83% |
| GLM, reduced feature set | 16.67% |
| GPR Bayesian | **13.89**% |
| GPR crossval | 22.22% |
| GPC | 23.61% |
| SVM linear | 22.22% |
| SVM linear, reduced feature set | 16.67% |
| SVM Gaussian | 20.83% |

Table 1: Error rates of different classification methods on the rheumatoid arthritis prediction problem. All error rates have been computed by 36-fold cross-validation. "Reduced feature set" indicates experiments where *a priori* feature selection has been done

counts $b$ and $c$ as the sufficient statistics of a binomial random variable with parameter $\theta$, where $\theta$ is the proportion of cases where method A performs better than method B.

The null hypothesis $H_0$ is that the parameter $\theta = 0.5$, that is, both methods A and B have the same performance. Hypothesis $H_1$ is that $\theta > 0.5$. The test statistics under the null hypothesis is the Binomial distribution $\text{Bi}(i|b+c,\theta)$ with parameter $\theta = 0.5$. We reject the null hypothesis if the probability of observing a count $k \geq c$ under the null hypothesis $P(k \geq c) = \sum_{i=c}^{b+c} \text{Bi}(i|b+c, \theta = 0.5)$ is sufficiently small.

**ROC Curves**   In medical diagnosis, biometrics and other areas, the common means of assessing a classification method is the receiver operating characteristics (ROC) curve. An ROC curve plots sensitivity versus 1-specificity[3] for different thresholds of the classifier output. Based on the ROC curve it can be decided how many false positives resp. false negatives one is willing to tolerate, thus helping to tune the classifier threshold to best suit a certain application.

Acquiring the ROC curve typically requires the classifier output on an independent test set. We instead use the union of all test set outputs in the cross-validation routine. This means that the ROC curve is based on outputs of slightly different models, yet this still seems to be the most suitable solution for such few data. For all classifiers we assess the area of the ROC curve and the cross-validation error rate. Here the above mentioned threshold on the classifier output is chosen such that sensitivity equals specificity.

## 5   Results

Tab. 1 lists error rates for all methods listed in Sec. 3. Gaussian process regression (GPR Bayesian) with an error rate of $\approx 14\%$ clearly outperforms all other methods, which all achieve comparable error rates in the range of $20\ldots24\%$. We attribute the good performance of GPR to its inherent feature relevance detection, which is done by adapting the length scales $\theta_i$ in the covariance function Eq. (2), i.e. a large $\theta_i$ means that the $i$-th feature is essentially ignored.

Surprisingly, Gaussian process classification implemented with Markov chain Monte Carlo sampling (GPC) showed rather poor performance. We currently have no clear explanation for this fact. We found no indications of convergence problems, furthermore we achieved similar results with different sampling schemes.

In an additional experiment we wanted to find out if classification results could be improved

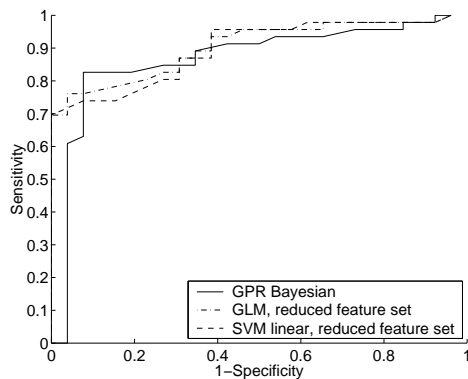

Figure 2: ROC curves of the best classification methods, both on the full data set and on a reduced data set where *a priori* feature selection was used to retain only the three most relevant features. Integrating the area under the ROC curves gives similar results for all three methods, with an area of 0.86 for SVM linear and GLM, and 0.84 for GPR Bayesian

by using only a subset of input features[4]. We found that only the performance of the two linear classifiers (GLM and SVM linear) could be improved by the input feature selection. Both now achieve an error rate of 16.67%, which is slightly worse than GPR on the full feature set (see Tab. 1).

**Significance Tests** Using the statistical hypothesis test described in the previous section, we compared all classification methods pairwise. It turned out the three best methods (GPR Bayesian, and GLM and SVM linear with reduced feature set) perform better than all other methods at a confidence level of 90% or more. Amongst the three best methods, no significant difference could be observed.

**ROC Curves** For the three best classification methods (GPR Bayesian, and GLM and SVM linear with reduced feature set), we have plotted the receiver operating characteristics (ROC) curve in Fig. 2. According to the ROC curve a sensitivity of $\approx 80\%$ can be achieved with a specificity at around 90%. GPR Bayesian seems to give best results, both in terms of error rate and shape of the ROC curve.

**Summary** To summarize, when the full set of features was used, best performance was obtained with GPR Bayesian. We attribute this to the inherent input relevance detection mechanisms of this approach. Comparable yet slightly worse results could be achieved by performing feature selection *a priori* and reducing the number of input features to the three most significant ones. In particular, the error rates of linear classifiers (GLM and linear SVM) improved by this feature selection, whereas more complex classifiers did not benefit. We can draw the important conclusion that, using the best classifiers, a sensitivity of 80% can be reached at a specificity of approximately 90%.

# 6   Conclusions

In this paper we have reported results of the analysis of a prototype medical imaging system, the RA scanner. Aim of the RA scanner is to detect soft tissue changes in finger joints,

which occur in early stages of rheumatoid arthritis (RA). Basis of the RA scanner is a novel laser imaging technique that is sensitive to inflammatory soft tissue changes.

We have analyzed whether the laser images are suitable for an accurate prediction of the inflammatory status of a finger joint, and which classification methods are best suited for this task. Out of a set of linear and kernel-based classification methods, Gaussian processes regression performed best, followed closely by generalized linear models and the linear support vector machine, the latter two operating on a reduced feature set. In particular, we have shown how parameter tuning and classifier training can be done on basis of the scarce available data. For the RA prediction task, we achieved a sensitivity of 80% at a specificity of approximately 90%. These results show that a further development of the RA scanner is desirable.

In the present study the inflammatory status is assessed by a rheumatologist, taking into account the patients subjective degree of pain. Thus we may expect a certain degree of label noise in the data we have trained the classification system on. Further developments of the classification system in the RA scanner will thus incorporate information from established medical imaging systems such as magnetic resonance imaging (MRI). MRI is known to provide accurate information about soft tissue changes in finger joints, yet is too costly to be routinely used for RA diagnosis. By incorporating MRI results into the RA scanner's classification system, we expect to significantly improve the overall accuracy.

**Acknowledgments** AS gratefully acknowledges support through an Ernst-von-Siemens scholarship. Thanks go to Radford Neal for making his FBM software available to the public, and to Ian Nabney and Chris Bishop for the Netlab toolbox.

## Footnotes

[1]As a prior distribution for kernel parameter $\theta_0$ we chose a Gamma distribution. $\theta_1 \ldots \theta_d$ are samples of a hierarchical Gamma distribution. In FBM syntax, the prior is `0.05:0.5 x0.2:0.5:1`. Sampling from the posterior distribution was done by persistent hybrid Monte Carlo, following the example of a 3-class problem in Neal [5].

[2]Thus, it is equivalent to a leave-one-out scheme, yet with only half the time consumption.

[3]$\text{sensitivity} = \frac{\text{true positives}}{\text{true positives} + \text{false negatives}}$   $\text{specificity} = \frac{\text{true negatives}}{\text{true negatives} + \text{false positives}}$

[4]This was done with the input relevance detection algorithm of the neural network tool SENN, a variant of sequential backward elimination where the feature that least affects the neural network output is removed. The feature set was reduced to the three most relevant ones.

### References

[1] Fahrmeir, L. and Tutz, G. *Multivariate Statistical Modelling Based on Generalized Linear Models*. Springer Verlag, 2nd edn., 2001.

[2] Kim, J. and Weisman, M. When does rheumatoid arthritis begin and why do we need to know? *Arthritis and Rheumatism*, 43:473–482, 2000.

[3] Lin, Y., Lee, Y., and Wahba, G. Support vector machines for classification in nonstandard situations. Tech. Rep. 1016, Department of Statistics, University of Wisconsin, Madison, WI, USA, 2000.

[4] Mika, S., Rätsch, G., Weston, J., Schölkopf, B., Smola, A. J., and Müller, K.-R. Invariant feature extraction and classification in kernel spaces. In S. A. Solla, T. K. Leen, and K.-R. Müller, eds., *Advances in Neural Information Processing Systems 12*. MIT Press, 2000.

[5] Neal, R. M. Monte carlo implementation of gaussian process models for bayesian regression and classification. Tech. Rep. 9702, Department of Statistics, University of Toronto, 1997.

[6] Prapavat, V., Runge, W., Krause, A., Beuthan, J., and Müller, G. A. Bestimmung von gewebeoptischen Eigenschaften eines Gelenksystems im Frühstadium der rheumatoiden Arthritis (in vitro). *Minimal Invasive Medizin*, 8:7–16, 1997.

[7] Scheel, A. K., Krause, A., Mesecke-von Rheinbaben, I., Metzger, G., Rost, H., Tresp, V., Mayer, P., Reuss-Borst, M., and Müller, G. A. Assessment of proximal finger joint inflammation in patients with rheumatoid arthritis, using a novel laser-based imaging technique. *Arthritis and Rheumatism*, 46(5):1177–1184, 2002.

[8] Schölkopf, B. and Smola, A. J. *Learning with Kernels*. MIT Press, 2002.

[9] Van Gestel, T., Suykens, J. A., Lanckriet, G., Lambrechts, A., De Moor, B., and Vandewalle, J. Bayesian framework for least-squares support vector machine classifiers, gaussian processes and kernel fisher discriminant analysis. *Neural Computation*, 14(5):1115–1147, 2002.

[10] Williams, C. K. and Barber, D. Bayesian classification with gaussian processes. *IEEE Transactions on Pattern Analysis and Machine Intelligence*, 20(12):1342–1351, 1998.

[11] Yang, Y. and Liu, X. A re-examination of text categorization methods. In *Proceedings of ACM SIGIR 1999*. ACM Press, 1999.